# Size of multilayer networks for exact learning: analytic approach

**André Elisseeff**
Dept Mathématiques et Informatique
École Normale Supérieure de Lyon
46 allée d'Italie
F69364 Lyon cedex 07, FRANCE

**Hélène Paugam-Moisy**
LIP, URA 1398 CNRS
École Normale Supérieure de Lyon
46 allée d'Italie
F69364 Lyon cedex 07, FRANCE

## Abstract

This article presents a new result about the size of a multilayer neural network computing real outputs for exact learning of a finite set of real samples. The architecture of the network is feedforward, with one hidden layer and several outputs. Starting from a fixed training set, we consider the network as a function of its weights. We derive, for a wide family of transfer functions, a lower and an upper bound on the number of hidden units for exact learning, given the size of the dataset and the dimensions of the input and output spaces.

## 1 RELATED WORKS

The context of our work is rather similar to the well-known results of Baum et al. [1, 2, 3, 5, 10], but we consider both real inputs and outputs, instead of the dichotomies usually addressed. We are interested in learning exactly all the examples of a fixed database, hence our work is different from stating that multilayer networks are universal approximators [6, 8, 9]. Since we consider real outputs and not only dichotomies, it is not straightforward to compare our results to the recent works about the VC-dimension of multilayer networks [11, 12, 13]. Our study is more closely related to several works of Sontag [14, 15], but with different hypotheses on the transfer functions of the units. Finally, our approach is based on geometrical considerations and is close to the model of Coetzee and Stonick [4].

First we define the model of network and the notations and second we develop our analytic approach and prove the fundamental theorem. In the last section, we discuss our point of view and propose some practical consequences of the result.

## 2 THE NETWORK AS A FUNCTION OF ITS WEIGHTS

General concepts on neural networks are presented in matrix and vector notations, in a geometrical perspective. All vectors are written in bold and considered as column vectors, whereas matrices are denoted with upper-case script.

### 2.1 THE NETWORK ARCHITECTURE AND NOTATIONS

Consider a multilayer network with $N_I$ input units, $N_H$ hidden units and $N_S$ output units. The inputs and outputs are real-valued. The hidden units compute a non-linear function $f$ which will be specified later on. The output units are assumed to be linear. A learning set of $N_P$ examples is given and fixed. For all $p \in \{1..N_P\}$, the $p^{th}$ example is defined by its input vector $\boldsymbol{d}_p \in \Re^{N_I}$ and the corresponding desired output vector $\boldsymbol{t}_p \in \Re^{N_S}$. The learning set can be represented as an *input matrix*, with both row and column notations, as follows

$$\mathcal{D} = \begin{bmatrix} d_{11} & d_{12} & \dots & d_{1N_I} \\ \vdots & \vdots & & \vdots \\ d_{N_P1} & d_{N_P2} & \dots & d_{N_PN_I} \end{bmatrix} = \begin{bmatrix} \boldsymbol{d}_1^T \\ \vdots \\ \boldsymbol{d}_{N_P}^T \end{bmatrix} = [\boldsymbol{\delta}_1, \dots, \boldsymbol{\delta}_{N_I}]$$

Similarly, the *target matrix* is $\mathcal{T} = \left[\boldsymbol{t}_1^T, \dots, \boldsymbol{t}_{N_P}^T\right]^T$, with independent row vectors.

### 2.2 THE NETWORK AS A FUNCTION $g$ OF ITS WEIGHTS

For all $h \in \{1..N_H\}$, $\boldsymbol{w}_h^1 = (w_{h1}^1, \dots, w_{hN_I}^1)^T \in \Re^{N_I}$ is the vector of the weights between all the input units and the $h^{th}$ hidden unit. The *input weight matrix* $W^1$ is defined as $W^1 = \left[\boldsymbol{w}_1^1, \dots, \boldsymbol{w}_{N_H}^1\right]$. Similarly, a vector $\boldsymbol{w}_s^2 = (w_{s1}^2, \dots, w_{sN_H}^2)^T \in \Re^{N_H}$ represents the weights between all the hidden units and the $s^{th}$ output unit, for all $s \in \{1..N_S\}$. Thus the *output weight matrix* $W^2$ is defined as $W^2 = \left[\boldsymbol{w}_1^2, \dots, \boldsymbol{w}_{N_S}^2\right]$. For an input matrix $\mathcal{D}$, the network computes an *output matrix*

$$\mathcal{Z}(\mathcal{D}) = \begin{bmatrix} z_1(\boldsymbol{d}_1) & \dots & z_{N_S}(\boldsymbol{d}_1) \\ \vdots & & \vdots \\ z_1(\boldsymbol{d}_{N_P}) & \dots & z_{N_S}(\boldsymbol{d}_{N_P}) \end{bmatrix} = \begin{bmatrix} \boldsymbol{z}(\boldsymbol{d}_1)^T \\ \vdots \\ \boldsymbol{z}(\boldsymbol{d}_{N_P})^T \end{bmatrix} = [\boldsymbol{\zeta}_1(\mathcal{D}), \dots, \boldsymbol{\zeta}_{N_S}(\mathcal{D})]$$

where each output vector $\boldsymbol{z}(\boldsymbol{d}_p)$ must be equal to the target $\boldsymbol{t}_p$ for exact learning. The network computation can be detailed as follows, for all $s \in \{1..N_S\}$

$$z_s(\boldsymbol{d}_p) = \sum_{h=1}^{N_H} w_{sh}^2 . f(\sum_{i=1}^{N_I} d_{pi}.w_{hi}^1)$$

$$= \sum_{h=1}^{N_H} w_{sh}^2 . f(\boldsymbol{d}_p^T.\boldsymbol{w}_h^1)$$

Hence, for the whole learning set, the $s^{th}$ output component is

$$\boldsymbol{\zeta}_s(\mathcal{D}) = \sum_{h=1}^{N_H} w_{sh}^2 . \begin{bmatrix} f(\boldsymbol{d}_1^T.\boldsymbol{w}_h^1) \\ \vdots \\ f(\boldsymbol{d}_{N_P}^T.\boldsymbol{w}_h^1) \end{bmatrix}$$

$$(1) \qquad \boldsymbol{\zeta}_s(\mathcal{D}) = \sum_{h=1}^{N_H} w_{sh}^2 . F(\mathcal{D}.\boldsymbol{w}_h^1)$$

In equation (1), $F$ is a vector operator which transforms a $n$ vector $v$ into a $n$ vector $F(v)$ according to the relation $[F(v)]_i = f([v]_i)$, $i \in \{1..n\}$. The same notation $F$ will be used for the matrix operator. Finally, the expression of the output matrix can be deduced from equation (1) as follows

$$\mathcal{Z}(\mathcal{D}) = \left[ F(\mathcal{D}.w_1^1), \ldots, F(\mathcal{D}.w_{N_H}^1) \right] \cdot \left[ w_1^2, \ldots, w_{N_S}^2 \right]$$

(2) $$\mathcal{Z}(\mathcal{D}) = F(\mathcal{D}.W^1).W^2$$

From equation (2), the network output matrix appears as a simple function of the input matrix and the network weights. Unlike Coetzee and Stonick, we will consider that the input matrix $\mathcal{D}$ is not a variable of the problem. Thus we express the network output matrix $\mathcal{Z}(\mathcal{D})$ as a function of its weights. Let $g$ be this function

$$g : \mathcal{R}^{N_I \times N_H + N_H \times N_S} \longrightarrow \mathcal{R}^{N_P \times N_S}$$
$$W = (W^1, W^2) \longrightarrow F(\mathcal{D}.W^1).W^2$$

The $g$ function clearly depends on the input matrix and could have be denoted by $g_\mathcal{D}$ but this index will be dropped for clarity.

## 3   FUNDAMENTAL RESULT

### 3.1   PROPERTY OF FUNCTION $g$

Learning is said to be exact on $\mathcal{D}$ if and only if there exists a network such that its output matrix $\mathcal{Z}(\mathcal{D})$ is equal to the target matrix $\mathcal{T}$. If $g$ is a diffeomorphic function from $R^{N_I \times N_H + N_H \times N_S}$ onto $R^{N_P \times N_S}$ then the network can learn any target in $R^{N_P \times N_S}$ exactly. We prove that it is sufficient for the network function $g$ to be a local diffeomorphism. Suppose there exist a set of weights $X$, an open subset $U \subset \mathcal{R}^{N_I N_H + N_H N_S}$ including $X$ and an open subset $V \subset \mathcal{R}^{N_P N_S}$ including $g(X)$ such that $g$ is diffeomorphic from $U$ to $V$. Since $V$ is an open neighborhood of $g(X)$, there exist a real $\lambda$ and a point $y$ in $V$ such that $\mathcal{T} = \lambda(y - g(X))$. Since $g$ is diffeomorphic from $U$ to $V$, there exists a set of weights $Y$ in $U$ such that $y = g(Y)$, hence $\mathcal{T} = \lambda(g(Y) - g(X))$. The output units of the network compute a linear transfer function, hence the linear combination of $g(X)$ and $g(Y)$ can be integrated in the output weights and a network with twice $N_I N_H + N_H N_S$ weights can learn $(\mathcal{D}, \mathcal{T})$ exactly (see Figure 1).

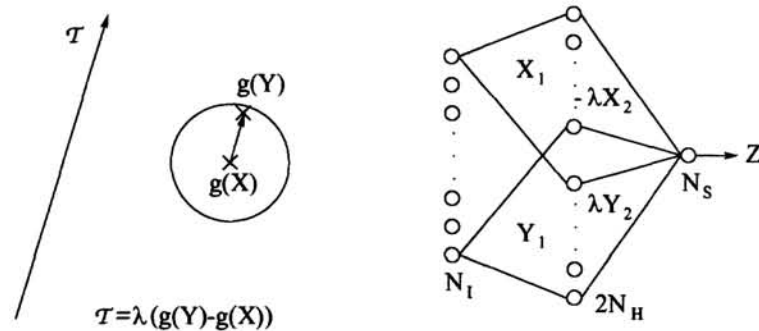

Figure 1: A network for exact learning of a target $\mathcal{T}$ (unique output for clarity)

For $g$ a local diffeomorphism, it is sufficient to find a set of weights $X$ such that the Jacobian of $g$ in $X$ is non-zero and to apply the theorem of local inversion. This analysis is developed in next sections and requires some assumptions on the transfer function $f$ of the hidden units. A function which verifies such an hypothesis $\mathcal{H}$ will be called a $\mathcal{H}$-function and is defined below.

## 3.2 DEFINITION AND THEOREM

**Definition 1** *Consider a function $f : \mathcal{R} \to \mathcal{R}$ which is $C^1(\mathcal{R})$ (i.e. with continuous derivative) and which has finite limits in $-\infty$ and $+\infty$. Such a function is called a $\mathcal{H}$-function iff it verifies the following property*

$$(\mathcal{H}) \qquad (\forall a \in \mathcal{R}/ \mid a \mid > 1) \quad \lim_{x \to \pm\infty} \mid \frac{f'(ax)}{f'(x)} \mid = 0$$

From this hypothesis on the transfer function of all the hidden units, the fundamental result can be stated as follows

**Theorem 1** *Exact learning of a set of $N_P$ examples, in general position, from $\mathcal{R}^{N_I}$ to $\mathcal{R}^{N_S}$, can be realized by a network with linear output units and a transfer function which is a $\mathcal{H}$-function, if the size $N_H$ of its hidden layer verifies the following bounds*

**Lower Bound** $\qquad N_H = \left\lceil \frac{N_P N_S}{N_I + N_S} \right\rceil \quad$ *hidden units are necessary*

**Upper Bound** $\qquad N_H = 2 \left\lceil \frac{N_P}{N_I + N_S} \right\rceil N_S \quad$ *hidden units are sufficient*

The proof of the lower bound is straightforward, since a condition for $g$ to be diffeomorphic from $R^{N_I \times N_H + N_H \times N_S}$ onto $R^{N_P \times N_S}$ is the equality of its input and output space dimensions $N_I N_H + N_H N_S = N_P N_S$.

## 3.3 SKETCH OF THE PROOF FOR THE UPPER BOUND

The $g$ function is an expression of the network as a function of its weights, for a given input matrix: $g(W^1, W^2) = F(\mathcal{D}.W^1).W^2$ and $g$ can be decomposed according to its vectorial components on the learning set (which are themselves vectors of size $N_S$). For all $p \in \{1..N_P\}$

$$g_p(W^1, W^2) = \boldsymbol{z}(\boldsymbol{d}_p) = \left[ \sum_{h=1}^{N_H} w_{1h}^2 \ f(\boldsymbol{d}_p^T.\boldsymbol{w}_h^1), \dots, \sum_{h=1}^{N_H} w_{N_S h}^2 \ f(\boldsymbol{d}_p^T.\boldsymbol{w}_h^1) \right]^T$$

The derivatives of $g$ w.r.t. the input weight matrix $W^1$ are, for all $i \in \{1..N_I\}$, for all $h \in \{1..N_H\}$

$$\frac{\partial g_p}{\partial w_{hi}^1} = \left[ w_{1h}^2 \ f'(\boldsymbol{d}_p^T.\boldsymbol{w}_h^1)d_{pi}, \dots, w_{N_S h}^2 \ f'(\boldsymbol{d}_p^T.\boldsymbol{w}_h^1)d_{pi} \right]^T$$

For the output weight matrix $W^2$, the derivatives of $g$ are, for all $h \in \{1..N_H\}$, for all $s \in \{1..N_S\}$

$$\frac{\partial g_p}{\partial w_{sh}^2} = [\ \underbrace{0, \dots, 0}_{s-1}, f(\boldsymbol{d}_p^T.\boldsymbol{w}_h^1), \underbrace{0, \dots, 0}_{N_S - s}\ ]^T$$

The Jacobian matrix $\mathcal{M}_J(g)$ of $g$, the size of which is $N_I N_H + N_H N_S$ columns and $N_S N_P$ rows, is thus composed of a block-diagonal part (derivatives w.r.t. $W^2$) and several other blocks (derivatives w.r.t. $W^1$). Hence the Jacobian $J(g)$ can be rewritten $J(g) = \mid J_1, J_2, \dots, J_{N_H} \mid$, after permutations of rows and columns, and using the Hadamard and Kronecker product notations, each $J_h$ being equal to

$$(3) \quad J_h = \left[ F(\mathcal{D}.\boldsymbol{w}_h^1) \otimes I_{N_S}, \left[ F'(\mathcal{D}.\boldsymbol{w}_h^1) \circ \delta_1 \dots F'(\mathcal{D}.\boldsymbol{w}_h^1) \circ \delta_{N_I} \right] \otimes \left[ w_{1h}^2 \dots w_{N_S h}^2 \right] \right]$$

where $I_{N_S}$ is for the identity matrix in dimension $N_S$.

Our purpose is to prove that there exists a point $X = (W^1, W^2)$ such that the Jacobian $J(g)$ is non-zero at $X$, i.e. such that the column vectors of the Jacobian matrix $\mathcal{M}_J(g)$ are linearly independent at $X$. The proof can be divided in two steps. First we address the case of a single output unit. Afterwards, this proof can be used to extend the result to several output units. Since the complete development of both proofs require a lot of calculations, we only present their sketchs below. More details can be found in [7].

### 3.3.1   Case of a single output unit

The proof is based on a linear arrangement of the projections of the column vectors of $J_h$ onto a subspace. This subspace is orthogonal to all the $J_i$ for $i < h$. We build a vector $w_h^1$ and a scalar $w_{1h}^2$ such that the projected column vectors are an independent family, hence they are independent with the $J_i$ for $i < h$. Such a construction is recursively applied until $h = N_H$. We derive then vectors $w_1^1, \ldots, w_{N_H}^1$ and $w_1^2$ such that $J(g)$ is non-zero. The assumption on $\mathcal{H}$-fonctions is essential for proving that the projected column vectors of $J_h$ are independent.

### 3.3.2   Case of multiple output units

In order to extend the result from a single output to $s$ output units, the usual idea consists in considering as many subnetworks as the number of output units. From this point of view, the bound on the hidden units would be $N_H = 2\frac{N_P N_S}{N_I + 1}$ which differs from the result stated in theorem 1. A new direct proof can be developed (see [7]) and get a better bound: the denominator is increased to $N_I + N_S$ .

## 4   DISCUSSION

The definition of a $\mathcal{H}$-function includes both sigmoids and gaussian functions which are commonly used for multilayer perceptrons and RBF networks, but is not valid for threshold functions. Figure 2 shows the difference between a sigmoid, which is a $\mathcal{H}$-function, and a saturation which is not a $\mathcal{H}$-function. Figures (a) and (b) represent the span of the output space by the network when the weights are varying, i.e. the image of $g$. For clarity, the network is reduced to 1 hidden unit, 1 input unit, 1 output unit and 2 input patterns. For a $\mathcal{H}$-function, a ball can be extracted from the output space $\mathcal{R}^2$, onto which the $g$ function is a diffeomorphism. For the saturation, the image of $g$ is reduced to two lines , hence $g$ cannot be onto on a ball of $\mathcal{R}^2$. The assumption of the activation function is thus necessary to prove that the jacobian is non-zero.

Our bound on the number of hidden units is very similar to Baum's results for dichotomies and functions from real inputs to binary outputs [1]. Hence the present result can be seen as an extension of Baum's results to the case of real outputs, and for a wide family of transfer functions, different from the threshold functions addressed by Baum and Haussler in [2]. An early result on sigmoid networks has been stated by Sontag [14]: for a single output and at least two input units, the number of examples must be twice the number of hidden units. Our upper bound on the number of hidden units is strictly lower than that (as soon as the number of input units is more than two). A counterpart of considering real data is that our results bear little relation to the VC-dimension point of view.

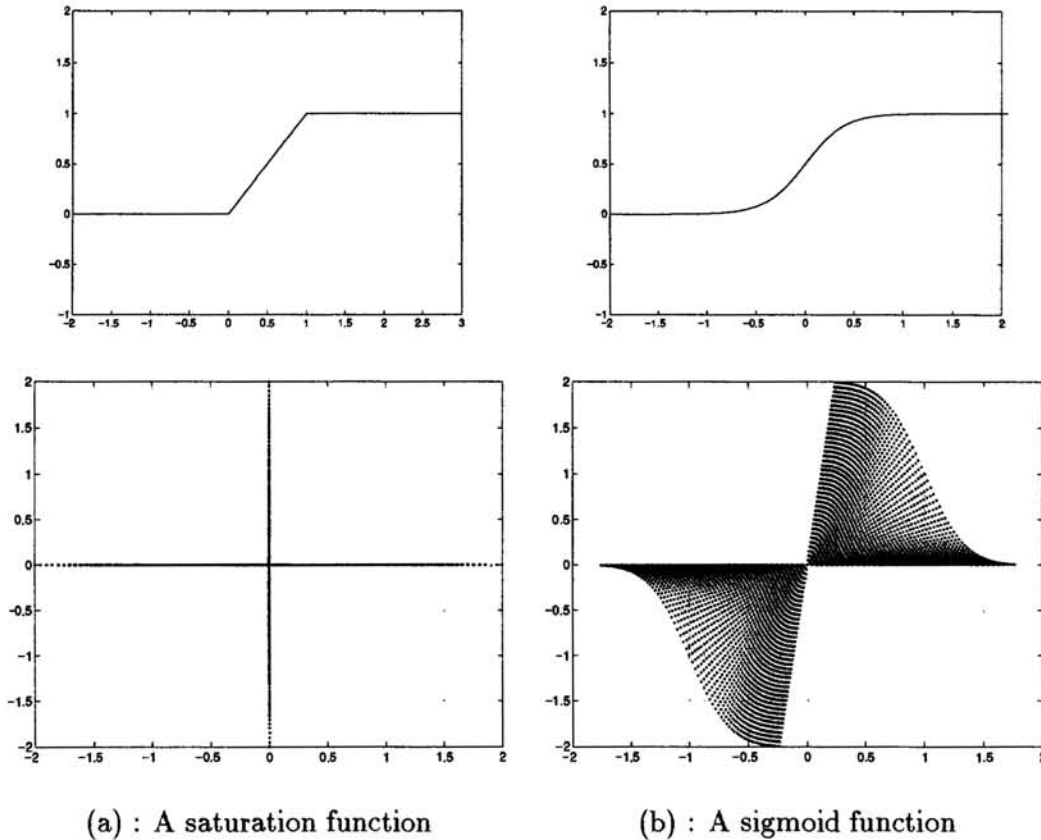

(a) : A saturation function    (b) : A sigmoid function

Figure 2: Positions of output vectors, for given data, when varying network weights

## 5   CONCLUSION

In this paper, we show that a number of hidden units $N_H = 2 \lceil N_P N_S / (N_I + N_S) \rceil$ is sufficient for a network of $\mathcal{H}$-functions to exactly learn a given set of $N_P$ examples in general position. We now discuss some of the practical consequences of this result.

According to this formula, the size of the hidden layer required for exact learning may grow very high if the size of the learning set is large. However, without *a priori* knowledge on the degree of redundancy in the learning set, exact learning is not the right goal in practical cases. Exact learning usually implies overfitting, especially if the examples are very noisy. Nevertheless, a right point of view could be to previously reduce the dimension and the size of the learning set by feature extraction or data analysis as pre-processing. Afterwards, our theoretical result could be a precious indication for scaling a network to perform exact learning on this representative learning set, with a good compromise between bias and variance.

Our bound is more optimistic than the rule-of-thumb $N_P = 10w$ derived from the theory of PAC-learning. In our architecture, the number of weights is $w = 2N_P N_S$. However the proof is not constructive enough to be derived as a learning algorithm, especially the existence of $g(Y)$ in the neighborhood of $g(X)$ where $g$ is a local diffeomorphism (cf. figure 1). From this construction we can only conclude that $N_H = \lceil N_P N_S / (N_I + N_S) \rceil$ is necessary and $N_H = 2 \lceil N_P N_S / (N_I + N_S) \rceil$ is sufficient to realize exact learning of $N_P$ examples, from $\mathcal{R}^{N_I}$ to $\mathcal{R}^{N_S}$.

The opportunity of using multilayer networks as auto-associative networks and for data compression can be discussed at the light of this results. Assume that $N_S = N_I$ and the expression of the number of hidden units is reduced to $N_H = N_P$ or at least $N_H = N_P/2$. Since $N_P \geq N_I + N_S$, the number of hidden units must verify $N_H \geq N_I$. Therefore, an architecture of "diabolo" network seems to be precluded for exact learning of auto-associations. A consequence may be that exact retrieval from data compression is hopeless by using internal representations of a hidden layer smaller than the data dimension.

## Acknowledgements

This work was supported by European Esprit III Project $n^0$ 8556, NeuroCOLT Working Group. We thank C.S. Poon and J.V. Shah for fruitful discussions.

## References

[1] E. B. Baum. On the capabilities of multilayer perceptrons. *J. of Complexity*, 4:193–215, 1988.

[2] E. B. Baum and D. Haussler. What size net gives valid generalization ? *Neural Computation*, 1:151–160, 1989.

[3] E. K. Blum and L. K. Li. Approximation theory and feedforward networks. *Neural Networks*, 4(4):511–516, 1991.

[4] F. M. Coetzee and V. L. Stonick. Topology and geometry of single hidden layer network, least squares weight solutions. *Neural Computation*, 7:672–705, 1995.

[5] M. Cosnard, P. Koiran, and H. Paugam-Moisy. Bounds on the number of units for computing arbitrary dichotomies by multilayer perceptrons. *J. of Complexity*, 10:57–63, 1994.

[6] G. Cybenko. Approximation by superpositions of a sigmoidal function. *Math. Control, Signal Systems*, 2:303–314, October 1988.

[7] A. Elisseeff and H. Paugam-Moisy. Size of multilayer networks for exact learning: analytic approach. Rapport de recherche 96-16, LIP, July 1996.

[8] K. Funahashi. On the approximate realization of continuous mappings by neural networks. *Neural Networks*, 2(3):183–192, 1989.

[9] K. Hornik, M. Stinchcombe, and H. White. Multilayer feedforward networks are universal approximators. *Neural Networks*, 2(5):359–366, 1989.

[10] S.-C. Huang and Y.-F. Huang. Bounds on the number of hidden neurones in multilayer perceptrons. *IEEE Trans. Neural Networks*, 2:47–55, 1991.

[11] M. Karpinski and A. Macintyre. Polynomial bounds for vc dimension of sigmoidal neural networks. In *27th ACM Symposium on Theory of Computing*, pages 200–208, 1995.

[12] P. Koiran and E. D. Sontag. Neural networks with quadratic vc dimension. In *Neural Information Processing Systems (NIPS*95)*, 1995. to appear.

[13] W. Maass. Bounds for the computational power and learning complexity of analog neural networks. In *25th ACM Symposium on Theory of Computing*, pages 335–344, 1993.

[14] E. D. Sontag. Feedforward nets for interpolation and classification. *J. Comp. Syst. Sci.*, 45:20–48, 1992.

[15] E. D. Sontag. Shattering all sets of $k$ points in "general position" requires *(k-1)/2* parameters. Technical Report Report 96-01, Rutgers Center for Systems and Control (SYCON), February 1996.
